# Dynamical Synapses Give Rise to a Power-Law Distribution of Neuronal Avalanches

**Anna Levina**[3,4]**, J. Michael Herrmann**[1,2]**, Theo Geisel**[1,2,4]
[1] Bernstein Center for Computational Neuroscience Göttingen
[2] Georg-August University Göttingen, Institute for Nonlinear Dynamics
[3] Graduate School Identification in Mathematical Models
[4] Max Planck Institute for Dynamics and Self-Organization
Bunsenstr. 10, 37073 Göttingen, Germany
anna|michael|geisel@chaos.gwdg.de

## Abstract

There is experimental evidence that cortical neurons show avalanche activity with the intensity of firing events being distributed as a power-law. We present a biologically plausible extension of a neural network which exhibits a power-law avalanche distribution for a wide range of connectivity parameters.

## 1 Introduction

Power-law distributions of event sizes have been observed in a number of seemingly diverse systems such as piles of granular matter [8], earthquakes [9], the game of life [1], friction [7], and sound generated in the lung during breathing. Because it is unlikely that the specific parameter values at which the critical behavior occurs are assumed by chance, the question arises as to what mechanisms may tune the parameters towards the critical state. Furthermore it is known that criticality brings about optimal computational capabilities [10], improves mixing or enhances the sensitivity to unpredictable stimuli [5]. Therefore, it is interesting to search for mechanisms that entail criticality in biological systems, for example in the nervous tissue.

In [6] a simple model of a fully connected neural network of non-leaky integrate-and-fire neurons was studied. This study not only presented the first example of a globally coupled system that shows criticality, but also predicted the critical exponent as well as some extra-critical dynamical phenomena, which were later observed in experimental researches. Recently, Beggs and Plenz [3] studied the propagation of spontaneous neuronal activity in slices of rat cortex and neuronal cultures using multi-electrode arrays. Thereby, they found avalanche-like activity where the avalanche sizes were distributed according to a power-law with an exponent of -3/2. This distribution was stable over a long period of time. The authors suggested that such a distribution is optimal in terms of transmission and storage of the information.

The network in [6] consisted of a set of $N$ identical threshold elements characterized by the membrane potential $u \geq 0$ and was driven by a slowly delivered random input. When the potential exceeds a threshold $\theta = 1$, the neuron spikes and relaxes. All connections

in the network are described by a single parameter $\alpha$ representing the evoked synaptic potential which a spiking neuron transmits to the all postsynaptic neurons. The system is driven by a slowly delivered random input. The simplicity of that model allows analytical consideration: an explicit formula for probability distribution of avalanche size depending on the parameter $\alpha$ was derived. A major drawback of the model was the lack of any true self-organization. Only at an externally well-tuned critical value of $\alpha = \alpha_{\mathrm{cr}}$ did the distribution take a form of a power-law, although with an exponent of precisely -3/2 (in the limit of a large system). The term *critical* will be applied here also to finite systems. True criticality requires a thermodynamic limit $N \longrightarrow \infty$, we consider approximate power-law behavior characterized by an exponent and an error that describes the remaining deviation from the best-matching exponent. The model in [6] is displayed for comparison in Fig. 3. In Fig. 1 (a-c) it is visible that the system may also exhibit other types of behavior such as small avalanches with a finite mean (even in the thermodynamic limit) at $\alpha < \alpha_{\mathrm{cr}}$. On the other hand at $\alpha > \alpha_{\mathrm{cr}}$ the distribution becomes non-monotonous, which indicates that avalanches of the size of the system are occurring frequently. Generally speaking, in order to drive the system towards criticality it therefore suffices to decrease the large avalanches and to enhance the small ones. Most interestingly, synaptic connections among real neurons show a similar tendency which thus deserves further study. We will consider the standard model of a short-term dynamics in synaptic efficacies [11, 13] and thereafter discuss several numerically determined quantities. Our studies imply that dynamical synapses indeed may support the criticalization of the neural activity in a small homogeneous neural system.

## 2   The model

We are considering a network of integrate-and-fire neurons with dynamical synapses. Each synapse is described by two parameters: amount of available neurotransmitters and a fraction of them which is ready to be used at the next synaptic event. Both parameters change in time depending on the state of the presynaptic neuron. Such a system keeps a long memory of the previous events and is known to exert a regulatory effect to the network dynamics, which will turnout to be beneficial.

Our approach is based on the model of dynamical synapses, which was shown by Tsodyks and Markram to reliably reproduce the synaptic responses between pyramidal neurons [11, 13]. Consider a set of $N$ integrate-and-fire neurons characterized by a membrane potential $h_i \geq 0$, and two connectivity parameters for each synapse: $J_{i,j} \geq 0$, $u_{i,j} \in [0,1]$. The parameter $J_{i,j}$ characterizes the number of available vesicles on the presynaptic side of the connection from neuron $j$ to neuron $i$. Each spike leads to the usage of a portion of the resources of the presynaptic neuron, hence, at the next synaptic event less transmitters will be available i.e. activity will be depressed. Between spikes vesicles are slowly recovering on a timescale $\tau_1$. The parameter $u_{i,j}$ denotes the actual fraction of vesicles on the presynaptic side of the connection from neuron $j$ to neuron $i$, which will be used in the synaptic transmission. When a spike arrives at the presynaptic side $j$, it causes an increase of $u_{i,j}$. Between spikes, $u_{i,j}$ slowly decrease to zero on a timescale $\tau_2$. The combined effect of $J_{i,j}$ and $u_{i,j}$ results in the facilitation or depression of the synapse. The dynamics of a membrane potential $h_i$ consists of the integration of excitatory postsynaptic currents over all synapses of the neuron and the slowly delivered random input. When the membrane potential exceeds threshold, the neuron emits a spike and $h_i$ resets to a smaller value. The

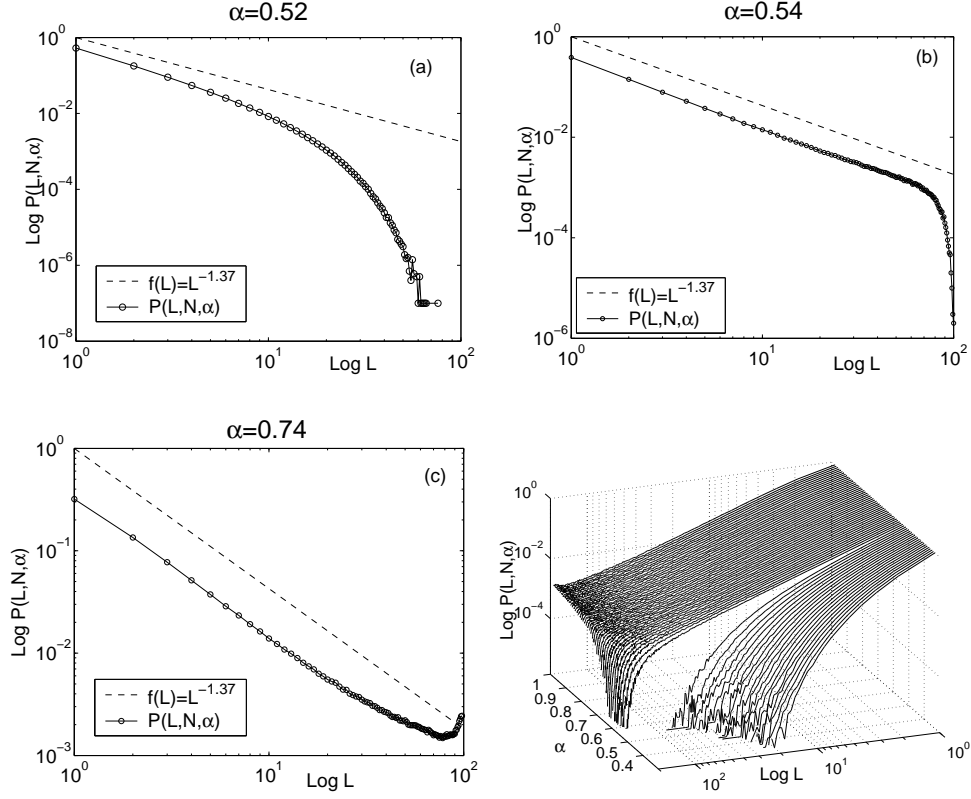

Figure 1: Probability distributions of avalanche sizes $P(L, N, \alpha)$. (a) in the subcritical, $\alpha = 0.52$, (b) the critical, $\alpha = 0.53$, and (c) supra-critical regime, $\alpha = 0.74$. In (a-c) the solid lines and symbols denote the numerical results for the avalanche size distributions, dashed lines show the best matching power-law. Here the curves are temporal averages over $10^6$ avalanches with $N = 100$, $u_0 = 0.1$, $\tau_1 = \tau_2 = 0.1$. Sub-figure (d) displays $P(L, N, \alpha)$ as a function of $L$ for $\alpha$ varying from 0.34 to 0.98 with step 0.01. The presented curves are temporal averages over $10^6$ avalanches with $N = 200$, $u_0 = 0.1$, $\tau_1 = \tau_2 = 0.1$.

joint dynamics can be written as a system of differential equations

$$\dot{J}_{i,j} = \frac{1}{\tau_1 \tau_s}(J_0 - J_{i,j}) - u_{i,j}J_{i,j}\delta(t - t_{\mathrm{sp}}^j), \tag{1}$$

$$\dot{u}_{i,j} = -\frac{1}{\tau_2 \tau_s}u_{i,j} + u_0(1 - u_{i,j})\delta(t - t_{\mathrm{sp}}^j), \tag{2}$$

$$\dot{h}_i = \frac{1}{\tau_s}\delta(r(t) - i)c\xi + \sum_{j=1}^N u_{i,j}J_{i,j}\delta(t - t_{\mathrm{sp}}^j) \tag{3}$$

Here $\delta(t)$ is the Dirac delta-function, $t_{\mathrm{sp}}^j$ is the spiking time of neuron $j$, $J_0$ is the resting value of $J_{i,j}$, $u_0$ is the minimal value of $u_{i,j}$, and $\tau_s$ is a parameter separating time-scales of random input and synaptic events. In the following study we will use the discrete version of equations (1-3).

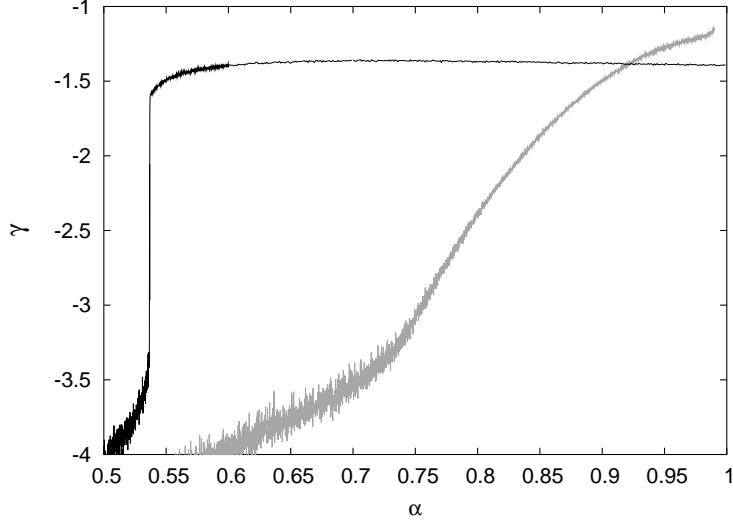

Figure 2: The best matching power-law exponent. The black line represents the present model, while the grey stands for model [6]. Average synaptic efficiency $\alpha$ varies from 0.3 to 1.0 with step 0.001. Presented curves are temporal averages over $10^7$ avalanches with $N = 200$, $u_0 = 0.1$, $\tau_1 = \tau_2 = 10$. Note that for a network of 200 units, the absolute critical exponent is smaller than the large system limit $\gamma = -1.5$ and that the step size has been drastically reduced in the vicinity of the phase transition.

## 3   Discrete version of the model

We consider time being measured in discrete steps, $t = 0, 1, 2, \ldots$. Because synaptic values are essentially determined presynaptically, we assume that all synapses of a neuron are identical, i.e. $J_j$, $u_j$ are used instead of $J_{i,j}$ and $u_{i,j}$ respectively. The system is initialized with arbitrary values $h_i \in [0, 1)$, $i = 1, \ldots, N$, where the threshold $\theta$ is fixed at 1. Depending on the state of the system at time $t$, the $i$-th element receives external input $I_i^{\text{ext}}(t)$ or internal input $I_i^{\text{int}}(t)$ from other neural elements. The two effects result in an activation $\tilde{h}$ at time $t + 1$,

$$\tilde{h}_i(t + 1) = h_i(t) + I_i^{\text{ext}}(t) + I_i^{\text{int}}(t) \tag{4}$$

From the activation $\tilde{h}_i(t + 1)$, the membrane potential of the $i$-th element at time $t + 1$ is computed as

$$h_i(t + 1) = \begin{cases} \tilde{h}_i(t + 1) & \text{if } \tilde{h}_i(t + 1) < 1, \\ \tilde{h}_i(t + 1) - 1 & \text{if } \tilde{h}_i(t + 1) \geq 1, \end{cases} \tag{5}$$

i.e. if the activation exceeds the threshold, it is reset but retains the supra-threshold portion $\tilde{h}_i(t + 1) - 1$ of the membrane potential.

The external input $I_i^{\text{ext}}(t)$ is a random amount $c\xi$, received by a randomly chosen neuron. Here, $c$ is input strength scale, parameter of the model, $\xi$ is uniformly distributed on $[0, 1]$ and independent of $i$. The external input is considered to be delivered slowly compared to the internal relaxation dynamics (which corresponds to $\tau_{\text{sep}} \gg 1$), i.e. it occurs only if no element has exceeded the threshold in the previous time step. This corresponds to an infinite separation of the time scales of external driving and avalanche dynamics discussed in the literature on self-organized criticality [12, 14]. The present results, however, are not affected by a continuous external input even during the avalanches. The external input

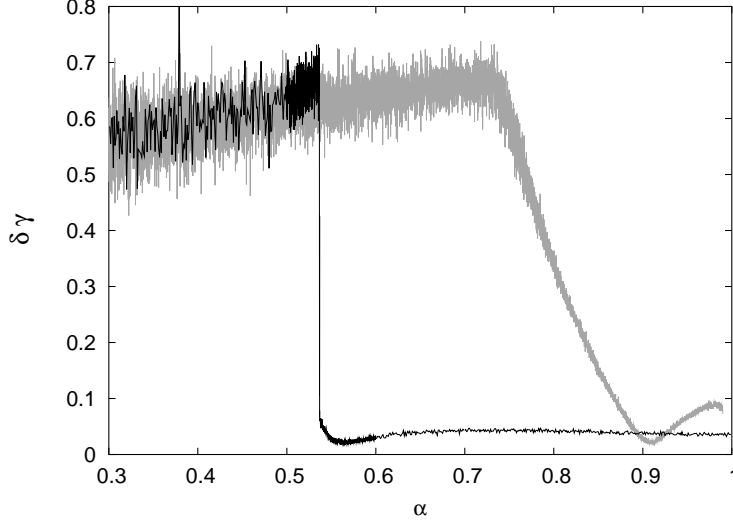

Figure 3: The mean squared deviation from the best fit power-law. The grey code and parameters are the same as in Fig. 2 For the fit, avalanches of a size larger than 1 and smaller than $N/2$ have been used. Clearly, an error levels above 0.1 indicates that the fitted curve is far from being a candidate for a power law. Near to $\alpha = 1$, when the non-dynamical model develops a supercritical behavior, the range of the power-law is quite limited. Interesting is again the sharp transition of the dynamical model, which is due to the facilitation strength surpassing a critical level.

can formally be written as $I_i^{\text{ext}}(t) = c\,\delta_{r,i}(t)\,|\delta_{M(t-1)|,\,0}\,\xi$, where $r$ is an integer random variable between 1 and $N$ indicating the chosen element, $M(t-1)$ is the set of indices of supra-threshold elements in the previous time step i.e. $M(t) = \{i|\tilde{h}_i(t) \geq 1\}$, and $\delta_{..}$ is the Kronecker delta. We will consider $c = J_0$, thus an external input is comparable with the typical internal input.

The internal input $I_i^{\text{int}}(t)$ is given by

$$I_i^{\text{int}}(t) = \sum_{j \in M(t-1)} J_j(t)\,u_j(t).$$

The system is initialized with $u_i = u_0$, $J_i = J_0$, where $J_0 = \alpha/(Nu_0)$ and $\alpha$ is the connection strength parameter. Similar to the membrane potentials dynamics, we can distinguish two situations: either there were supra-threshold neurons at the previous moment of time or not.

$$u_j(t+1) = \begin{cases} u_j(t) - \frac{1}{\tau_2}u_0 u_j(t)) \cdot \delta_{|M(t)|,0} & \text{if } \tilde{h}_i(t) < 1, \\ u_j(t) + (1 - u_j(t))u_0(t) & \text{if } \tilde{h}_i(t) \geq 1, \end{cases} \tag{6}$$

$$J_j(t+1) = \begin{cases} J_j(t) + \frac{1}{\tau_1}(J_0 - J_j(t)) \cdot \delta_{|M(t)|,0} & \text{if } \tilde{h}_i(t) < 1, \\ J_j(t)(1 - u_j(t)) & \text{if } \tilde{h}_i(t) \geq 1, \end{cases} \tag{7}$$

Thus, we have a model with parameters $\alpha$, $u_0$, $\tau_1$, $\tau_2$ and $N$. Our main focus will be on the influence of $\alpha$ on the cumulative dynamics of the network. The dependence on $N$ has been studied in [6], where it was found that the critical parameter of the distribution scales as $\alpha_{\text{cr}} = 1 - N^{-1/2}$. In the same way, the exponent will be smaller in modulus than -3/2 for finite systems.

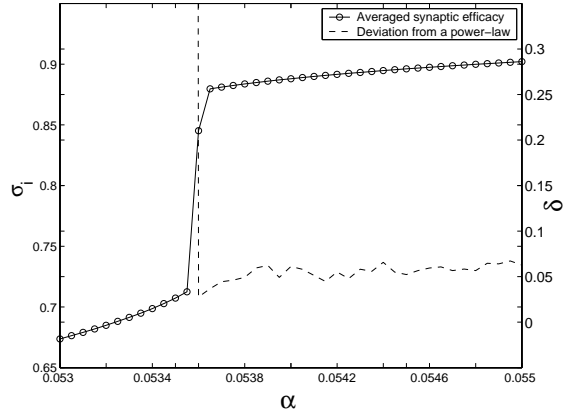

Figure 4: Average synaptic efficacy for the parameter $\alpha$ varied from 0.53 to 0.55 with step 0.0005 (left axis). Dashed line depicts deviation from a power-law (right axis).

If at time $t_0$ an element receives an external input and fires, then an avalanche starts and $|M(t_0)| = 1$. The system is globally coupled, such that during an avalanche all elements receive internal input including the unstable elements themselves. The avalanche duration $D \geq 0$ is defined to be the smallest integer for which the stopping condition $|M(t_0+D)| = 0$ is satisfied. The avalanche size $L$ is given by $L = \sum_{k=0}^{D-1} |M(t_0 + k)|$. The subject of our interest is the probability distribution of avalanche size $P(L, N, \alpha)$ depending on the parameter $\alpha$.

## 4 Results

Similarly, as in model [6] we considered the avalanche size distribution for different values of $\alpha$, cf. Fig. 1. Three qualitatively different regimes can be distinguished: subcritical, critical, and supra-critical. For small values of $\alpha$, subcritical avalanche-size distributions are observed. The subcriticality is characterized by the neglible number of avalanches of a size close to the system size. For $\alpha_{\mathrm{cr}}$, the system has an avalanche distribution with an approximate power-law behavior for $L$, inside a range from 1 almost up to the size of the system, where the exponential cut-off is observed (Fig. 1b). Above the critical value $\alpha_{\mathrm{cr}}$, avalanche size distributions become non-monotonous (Fig. 1c). Such supra-critical curves have a minimum at an intermediate avalanche size.

There is the sharp transition from subcritical to critical regime and then a long *critical region*, where the distribution of avalanche size stays close to the power-law. For a system of 200 neurons this transition is shown in Fig. 2. To characterize this effect we used the least-squares estimate of the closest power-law parameters $C_{\mathrm{norm}}$ and $\gamma$.

$$p(L, N, \alpha) \approx C_{\mathrm{norm}} L^{\gamma}$$

The mean squared deviation from the estimated power-law undergoes a fast change Fig. 3 (bottom) near $\alpha_{\mathrm{cr}} = 0.54$. At this point the transition from the subcritical to the critical regime occurs. Then there is a long interval of parameters for which the deviation from the power-law is about 2%. Also, the parameters of the power-law approximately stay constant. For different system-sizes different values of $\alpha_{\mathrm{cr}}$ and $\gamma$ are observed. At large system sizes $\gamma$ is close to $-1.5$

In order to develop more extensive analysis we considered also a number of additional sta-

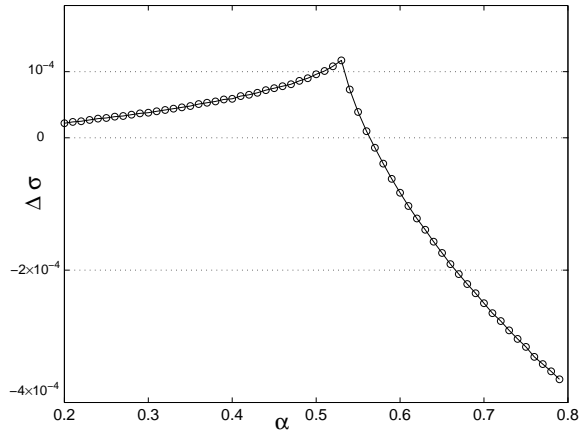

Figure 5: Difference between synaptic efficacy after and before avalanche averaged over all synapses . Values larger than zero mean facilitation, smaller ones mean depression. Presented curves are temporal averages over $10^6$ avalanches with $N = 100$, $u_0 = 0.1$, $\tau_1 = \tau_2 = 10$.

tistical quantities at the beginning and after the avalanche. The average synaptic efficacy $\sigma = \langle \sigma_i \rangle = \langle J_i u_i \rangle$ is determined by taking the average over all neurons participating in an avalanche. This average shows the mean input, which neurons receive at each step of avalanche. This characteristic quantity undergoes a sharp transition together with the avalanches distribution, cf. Fig. 4. The meaning of the quantity $\sigma$ in the present model is similar to the coupling strength $\alpha/N$ in the model discussed in [6]. It is equal to the average EPSP which all postsynaptic neurons will receive after presynaptic neuron spikes. The transition from a subcritical to a critical regime happens when $\sigma$ jumps into the vicinity of $\alpha_{\mathrm{cr}}/N$ of the previous model (for $N = 100$ and $\alpha_{\mathrm{cr}} = 0.9$). This points to the correspondence between the two models.

When $\alpha$ is large, then the synaptic efficacy is high and, hence, avalanches are large and intervals between them are small. The depression during the avalanche dominates facilitation and decrease synaptic efficacy and vise versa. When avalanches are small, facilitation dominates depression. Thus, the synaptic dynamics stabilizes the network to remain near the critical value for a large interval of parameters $\alpha$. In Fig. 4 shown the averaged effect of an avalanche for different values of parameter $\alpha$. For $\alpha > \alpha_{\mathrm{cr}}$, depression during the avalanche is stronger than facilitation and avalanches on average decrease synaptic efficacy. When $\alpha$ is very small, the effect of facilitation is washed out during the inter-avalanche period where synaptic parameters return to the resting state. To illustrate this, Fig. 5 shows the difference, $\Delta\sigma = \langle \sigma_{\mathrm{after}} \rangle - \langle \sigma_{\mathrm{before}} \rangle$, between the average synaptic efficacies after and before the avalanche depending on the parameter $\alpha$. If this difference is larger than zero, synapses are facilitated by avalanche. If it is smaller than zero, synapses are depressed. For small values of the parameter $\alpha$ avalanches lead to facilitation, while, for large values of $\alpha$ avalanches depress synapses.

In the limit $N \to \infty$, the synaptic dynamics should be rescaled such that the maximum of transmitter available at a time $t$ divided by the average avalanche size converges to a value which scales as $1 - N^{-1/2}$. In this way, if the average avalanche size is smaller than critical, synapses will essentially be enhanced, or they will otherwise experience depression. The necessary parameters for the model (such as the time-scales) have shown to be easily achievable in the small (although time-consuming) simulations presented here.

# 5  Conclusion

We presented a simple biologically plausible complement to a model of a non-leaky integrate-and-fire neurons network which exhibits a power-law avalanche distribution for a wide range of connectivity parameters. In previous studies [6] we showed, that the simplest model with only one parameter $\alpha$, characterizing synaptic efficacy of all synapses exhibits subcritical, critical and supra critical regimes with continuous transition from one to another, depending on parameter $\alpha$. These main classes are also present here but the region of critical behavior is immensely enlarged. Both models have a power-law distribution with an exponent approximately equal to -3/2, although the exponent is somewhat smaller for small network sizes. For network sizes close to those in the experiments described in [3] the result is indistinguishable from the limiting value.

# References

[1] P. Bak, K. Chen, and M. Creutz. Self-organized criticality in the 'Game of Life. *Nature*, 342:780–782, 1989.

[2] P. Bak, C. Tang, and K. Wiesenfeld. Self-organized criticality: an explanation of $1/f$ noise. *Phys. Rev. Lett.*, 59:381–384, 1987.

[3] J. Beggs and D. Plenz. Neuronal avalanches in neocortical circuits. *J Neurosci*, 23:11167–11177, 2003.

[4] J. Beggs and D. Plenz. Neuronal Avalanches Are Diverse and Precise Activity Patterns That Are Stable for Many Hours in Cortical Slice Cultures. *J Neurosci*, 24(22):5216-5229, 2004.

[5] R. Der, F. Hesse, R. Liebscher ( Contingent robot behavior from self-referential dynamical systems. Submitted to *Autonomous Robots*, 2005.

[6] C. W. Eurich, M. Herrmann, and U. Ernst. Finite-size effects of avalanche dynamics. *Phys. Rev. E*, 66, 2002.

[7] H. J. S. Feder and J. Feder. Self-organized criticality in a stick-slip process. *Phys. Rev. bibtLett.*, 66:2669–2672, 1991.

[8] V. Frette, K. Christensen, A. M. Malthe-Sørenssen, J. Feder, T. Jøssang, and P. Meakin. Avalanche dynamics in a pile of rice. *Nature*, 397:49, 1996.

[9] B. Gutenberg and C. F. Richter. Magnitude and energy of earthquakes. *Ann. Geophys.*, 9:1, 1956.

[10] R. A. Legenstein, W. Maass. Edge of chaos and prediction of computational power for neural microcircuit models. Submitted, 2005.

[11] H. Markram and M. Tsodyks. Redistribution of synaptic efficacy between pyramidal neurons. *Nature*, 382:807–810, 1996.

[12] D. Sornette, A. Johansen, and I. Dornic. Mapping self-organized criticality onto criticality. *J. Phys. I*, 5:325–335, 1995.

[13] M. Tsodyks, K. Pawelzik, and H. Markram. Neural networks with dynamic synapses. *Neural Computations*, 10:821–835, 1998.

[14] A. Vespignani and S. Zapperi. Order parameter and scaling fields in self-organized criticality. *Phys. Rev. Lett.*, 78:4793–4796, 1997.
